# Multi-modular Associative Memory

**Nir Levy    David Horn**
School of Physics and Astronomy
Tel-Aviv University Tel Aviv 69978, Israel

**Eytan Ruppin**
Departments of Computer Science & Physiology
Tel-Aviv University Tel Aviv 69978, Israel

## Abstract

Motivated by the findings of modular structure in the association cortex, we study a multi-modular model of associative memory that can successfully store memory patterns with different levels of activity. We show that the segregation of synaptic conductances into intra-modular linear and inter-modular nonlinear ones considerably enhances the network's memory retrieval performance. Compared with the conventional, single-module associative memory network, the multi-modular network has two main advantages: It is less susceptible to damage to columnar input, and its response is consistent with the cognitive data pertaining to category specific impairment.

## 1    Introduction

Cortical modules were observed in the somatosensory and visual cortices a few decades ago. These modules differ in their structure and functioning but are likely to be an elementary unit of processing in the mammalian cortex. Within each module the neurons are interconnected. Input and output fibers from and to other cortical modules and subcortical areas connect to these neurons. More recently, modules were also found in the association cortex [1] where memory processes supposedly take place. Ignoring the modular structure of the cortex, most theoretical models of associative memory have treated single module networks. This paper develops a novel multi-modular network that mimics the modular structure of the cortex. In this framework we investigate the computational rational behind cortical multi-modular organization, in the realm of memory processing.

Does multi-modular structure lead to computational advantages? Naturally one

may think that modules are necessary in order to accommodate memories of different coding levels. We show in the next section that this is not the case, since one may accommodate such memories in a standard sparse coding network . In fact, when trying to capture the same results in a modular network we run into problems, as shown in the third section: If both inter and intra modular synapses have linear characteristics, the network can sustain memory patterns with only a limited range of activity levels. The solution proposed here is to distinguish between intra-modular and inter-modular couplings, endowing the inter-modular ones with nonlinear characteristics. From a computational point of view, this leads to a modular network that has a large capacity for memories with different coding levels. The resulting network is particularly stable with regard to damage to modular inputs. From a cognitive perspective it is consistent with the data concerning category specific impairment.

## 2 Homogeneous Network

We study an excitatory-inhibitory associative memory network [2], having $N$ excitatory neurons. We assume that the network stores $M_1$ memory patterns $\eta^\mu$ of sparse coding level $p$ and $M_2$ patterns $\xi^\nu$ with coding level $f$ such that $p < f << 1$. The synaptic efficacy $J_{ij}$ between the $j$th (presynaptic) neuron and the $i$th (postsynaptic) neuron is chosen in the Hebbian manner

$$J_{ij} = \frac{1}{Np} \sum_{\mu=1}^{M_1} \eta^\mu{}_i \eta^\mu{}_j + \frac{1}{Np} \sum_{\mu=1}^{M_2} \xi^\nu{}_i \xi^\nu{}_j \,, \tag{1}$$

The updating rule for the activity state $V_i$ of the $i$th binary neuron is given by

$$V_i(t+1) = \Theta\left(h_i(t) - \theta\right) \tag{2}$$

where $\Theta$ is the step function and $\theta$ is the threshold.

$$h_i(t) = h_i^e(t) - \frac{\gamma}{p} \mathcal{Q}(t) \tag{3}$$

is the local field, or membrane potential. It includes the excitatory Hebbian coupling of all other excitatory neurons,

$$h_i^e(t) = \sum_{j \neq i}^{N} J_{ij} V_j(t) \,, \tag{4}$$

and global inhibition that is proportional to the total activity of the excitatory neurons

$$\mathcal{Q}(t) = \frac{1}{N} \sum_{j}^{N} V_j(t) \,. \tag{5}$$

The overlap $m(t)$ between the network activity and the memory patterns is defined for the two memory populations as

$$m_\xi{}^\nu(t) = \frac{1}{Nf} \sum_{j}^{N} \xi^\nu{}_j V_j(t) \,, \qquad m_\eta{}^\mu(t) = \frac{1}{Np} \sum_{j}^{N} \eta^\mu{}_j V_j(t) \,. \tag{6}$$

The storage capacity $\alpha = M/N$ of this network has two critical capacities. $\alpha_{c\xi}$ above which the population of $\xi^\nu$ patterns is unstable and $\alpha_{c\eta}$ above which the population of $\eta^\mu$ patterns is unstable. We derived equations for the overlap and total activity of the two populations using mean field analysis. Here we give the

fixed-point equations for the case of $M_1 = M_2 = \frac{M}{2}$ and $\gamma = M_1 f^2 + M_2 p^2$. The resulting equations are

$$m_\eta = \Phi\left(\frac{\theta - m_\eta}{\phi}\right) , \qquad\qquad \mathcal{Q} = p m_\eta + \Phi\left(\frac{\theta}{\phi}\right) , \qquad (7)$$

and

$$m_\xi = \Phi\left(\frac{\theta - \frac{f}{p} m_\xi}{\phi}\right) , \qquad\qquad \mathcal{Q} = f m_\xi + \Phi\left(\frac{\theta}{\phi}\right) , \qquad (8)$$

where

$$\phi^2 = \frac{1}{2}\alpha \mathcal{Q}\left(1 + \frac{f^2}{p^2}\right) + \frac{1}{2}\alpha N p \mathcal{Q}^2\left(1 + \frac{f^3}{p^3}\right) , \qquad (9)$$

and

$$\Phi(x) = \int_x^\infty \exp\left(-\frac{z^2}{2}\right)\frac{dz}{\sqrt{2\pi}} . \qquad (10)$$

(a)                                                                          (b)

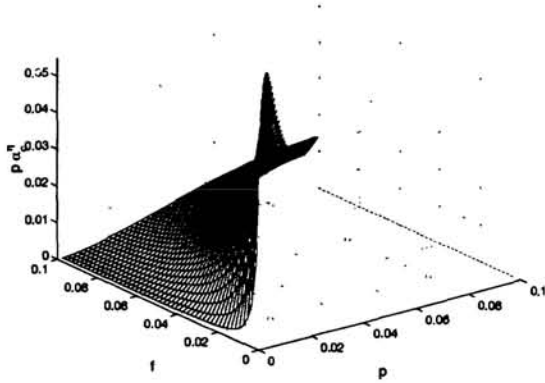
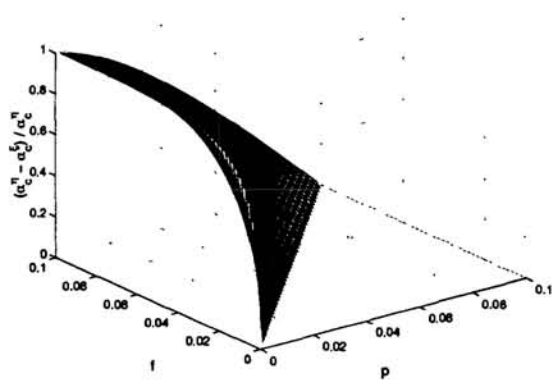

Figure 1: (a) The critical capacity $\alpha_{c\eta}$ vs. $f$ and $p$ for $f \geq p$, $\theta = 0.8$ and $N = 1000$. (b) $(\alpha_{c\eta} - \alpha_{c\xi})/\alpha_{c\eta}$ versus $f$ and $p$ for the same parameters as in (a). The validity of these analytical results was tested and verified in simulations.

Next, we look for the critical capacities, $\alpha_{c\eta}$ and $\alpha_{c\xi}$ at which the fixed-point equations become marginally stable. The results are shown in Figure 1. Figure 1(a) shows $\alpha_{c\eta}$ vs. the coding levels $f$ and $p$ ($f \geq p$). Similar results were obtained for $\alpha_{c\xi}$. As evident the critical capacities of both populations are smaller than the one observed in a homogeneous network in which $f = p$. One hence necessarily pays a price for the ability to store patterns with different levels of activity.

Figure 1(b) plots the relative capacity difference $(\alpha_{c\eta} - \alpha_{c\xi})/\alpha_{c\eta}$ vs. $f$ and $p$. The function is non negative, i.e., $\alpha_{c\eta} \geq \alpha_{c\xi}$ for all $f$ and $p$. Thus, low activity memories are more stable than high activity ones.

Assuming that high activity codes more features [3], these results seem to be at odds with the view [3, 4] that memories that contain more semantic features, and therefore correspond to larger Hebbian cell assemblies, are more stable, such as concrete versus abstract words. The homogeneous network, in which the memories with high activity are more susceptible to damage, cannot account for these observations. In the next section we show how a modular network can store memories with different activity levels and account for this cognitive phenomenon.

## 3   Modular Network

We study a multi modular excitatory-inhibitory associative memory network, storing $M$ memory patterns in $L$ modules of $N$ neurons each. The memories are coded such that in every memory a variable number $\Omega$ of 1 to $L$ modules is active. This number will be denoted as *modular coding*. The coding level inside the modules is sparse and fixed, i.e., each modular Hebbian cell assembly consists of $pN$ active neurons with $p << 1$. The synaptic efficacy $J_{ij}{}^{lk}$ between the $j$th (presynaptic) neuron from the $k$th module and the $i$th (postsynaptic) neuron from the $l$th module is chosen in a Hebbian manner

$$J_{ij}{}^{lk} = \frac{1}{Np} \sum_{\mu=1}^{M} \eta^{\mu}{}_{il} \eta^{\mu}{}_{jk} \; , \tag{11}$$

where $\eta^{\mu}{}_{il}$ are the stored memory patterns. The updating rule for the activity state $V_i{}^l$ of the $i$th binary neuron in the $l$th module is given by

$$V_i{}^l(t+1) = \mathcal{S}\left( h_i{}^l(t) - \theta_s \right) \; , \tag{12}$$

where $\theta_s$ is the threshold, and $\mathcal{S}(x)$ is a stochastic sigmoid function, getting the value 1 with probability $(1 + e^{-x})^{-1}$ and 0 otherwise. The neuron's local field, or membrane potential has two components,

$$h_i{}^l(t) = h_i{}^l{}_{internal}(t) + h_i{}^l{}_{external}(t) \; . \tag{13}$$

The internal field, $h_i{}^l{}_{internal}(t)$, includes the contributions from all other excitatory neurons that are situated in the $l$th module, and inhibition that is proportional to the total modular activity of the excitatory neurons, i.e.,

$$h_i{}^l{}_{internal}(t) = \sum_{j \neq i}^{N} J_{ij}{}^{ll} V_j{}^l(t) - \gamma_s \mathcal{Q}^l(t) \; , \tag{14}$$

where

$$\mathcal{Q}^l(t) = \frac{1}{Np} \sum_{j}^{N} V_j{}^l(t) \; . \tag{15}$$

The external field component, $h_i{}^l{}_{external}(t)$, includes the contributions from all other excitatory neurons that are situated outside the $l$th module, and inhibition that is proportional to the total network activity.

$$h_i{}^l{}_{external}(t) = \mathcal{G}\left( \sum_{k \neq l}^{L} \sum_{j}^{N} J_{ij}{}^{lk} V_j{}^k(t) - \gamma_d \sum_{k}^{L} \mathcal{Q}^k(t) - \theta_d \right) \; . \tag{16}$$

We allow here for the freedom of using more complicated behavior than the standard $\mathcal{G}(x) = x$ one. In fact, as we will see, the linear case is problematic, since only memory storage with limited modular coding is possible.

The retrieval quality at each trial is measured by the overlap function, defined by

$$m^{\mu}(t) = \frac{1}{pN\Omega^{\mu}} \sum_{k=1}^{L} \sum_{i=1}^{N} \eta^{\mu}{}_{ik} V_i{}^k(t) \; , \tag{17}$$

where $\Omega^{\mu}$ is the modular coding of $\eta^{\mu}$.

In the simulations we constructed a network of $L = 10$ modules, where each module contains $N = 500$ neurons. The network stores $M = 50$ memory patterns randomly distributed over the modules. Five sets of ten memories each are defined. In each set the modular coding is distributed homogeneously between one to ten active modules. The sparse coding level within each module was set to be $p = 0.05$. Every simulation experiment is composed of many trials. In each trial we use as initial condition a corrupted version of a stored memory pattern with error rate of 5%, and check the network's retrieval after it converges to a stable state.

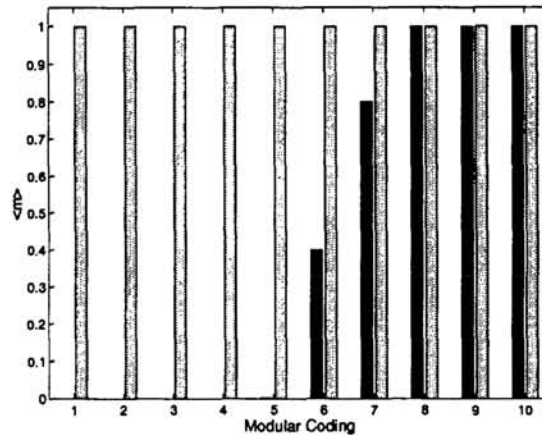

Figure 2: Quality of retrieval *vs.* memory modular coding. The dark shading represents the mean overlap achieved by a network with linear intra-modular and inter-modular synaptic couplings. The light shading represents the mean overlap of a network with sigmoidal inter-modular connections, which is perfect for all memory patterns. The simulation parameters were: $L = 10$, $N = 500$, $M = 50$, $p = 0.05$, $\lambda = 0.7$, $\theta_d = 2$ and $\theta_s = 0.6$.

We start with the standard choice of $\mathcal{G}(x) = x$, i.e. treating similarly the intra-modular and inter-modular synaptic couplings. The performance of this network is shown in Figure 2. As evident, the network can store only a relatively narrow span of memories with high modular coding levels, and completely fails to retrieve memories with low modular coding levels (see also [5]). If, however, $\mathcal{G}$ is chosen to be a sigmoid function, a completely stable system is obtained, with all possible coding levels allowed. A sigmoid function on the external connections is hence very effective in enhancing the span of modular coding of memories that the network can sustain. The segregation of the synaptic inputs to internal and external connections has been motivated by observed patterns of cortical connectivity: Axons forming excitatory intra-modular connections make synapses more proximal to the cell body than do inter-modular connections [6]. Dendrites, having active conductances, embody a rich repertoire of nonlinear electrical and chemical dynamics (see [7] for a review). In our model, the setting of $\mathcal{G}$ to be a sigmoid function crudely mimics these active conductance properties.

We may go on and envisage the use of a nested set of sigmoidal dendritic transmission functions. This turns out to be useful when we test the effects of pathologic alterations on the retrieval of memories with different modular codings. The amazing result is that if the damage is done to modular inputs, the highly nonlinear transmission functions are very resistible to it. An example is shown in Fig. 3.

Here we compare two nonlinear functions:

$$\mathcal{G}_1 = \lambda\Theta\left[\sum_{k\neq l}^{L}\sum_{j}^{N} J_{ij}{}^{lk} V_j{}^k(t) - \gamma_d \sum_{k\neq l}^{L} \mathcal{Q}_k(t) - \theta_d\right] \;,$$

$$\mathcal{G}_2 = \lambda\Theta\left[\sum_{k\neq l}^{L}\Theta\left[\sum_{j}^{N} J_{ij}{}^{lk} V_j{}^k(t) - \gamma_d \mathcal{Q}_k(t) - \theta_k\right] - \theta_d\right] \;.$$

The second one is the nested sigmoidal function mentioned above. Two types of input cues are compared: correct $\eta^\mu{}_{il}$ to one of the modules and no input to the rest, or partial input to all modules.

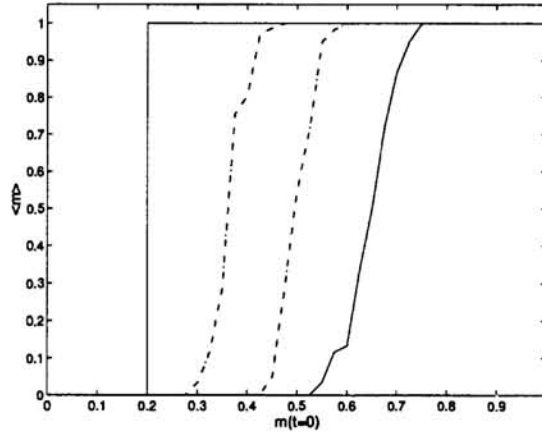

Figure 3: The performance of modular networks with different types of non-linear inter-connections when partial input cues are given. The mean overlap is plotted *vs.* the overlap of the input cue. The solid line represents the performance of the network with $\mathcal{G}_2$ and the dash-dot line represents $\mathcal{G}_1$. The left curve of $\mathcal{G}_2$ corresponds to the case when full input is presented to only one module (out of the 5 that comprise a memory), while the right solid curve corresponds to partial input to all modules. The two $\mathcal{G}_1$ curves describe partial input to all modules, but correspond to two different choices of the threshold parameter $\theta_d$, 1.5 (left) and 2 (right). Parameters are $L = 5$, $N = 1000$, $p = 0.05$, $\lambda = 0.8$, $\Omega = 5$, $\theta_s = 0.7$ and $\theta_k = 0.7$.

As we can see, the nested nonlinearities enable retrieval even if only the input to a single module survives. One may therefore conclude that, under such conditions, patterns of high modular coding have a grater chance to be retrieved from an input to a single module and thus are more resilient to afferent damage. Adopting the assumption that different modules code for distinct semantic features, we now find that a multi-modular network with nonlinear dendritic transmission can account for the view of [3], that memories with more features are more robust.

## 4 Summary

We have studied the ability of homogeneous (single-module) and modular networks to store memory patterns with variable activity levels. Although homogeneous networks can store such memory patterns, the critical capacity of low activity memories was shown to be larger than that of high activity ones. This result seems to be inconsistent with the pertaining cognitive data concerning category specific semantic

impairment, which seem to imply that high activity memories should be the more stable ones.

Motivated by the findings of modular structure in associative cortex, we developed a multi-modular model of associative memory. Adding the assumption that dendritic non-linear processing operates on the signals of inter-modular synaptic connections, we obtained a network that has two important features: coexistence of memories with different modular codings and retrieval of memories from cues presented to a small fraction of all modules. The latter implies that memories encoded in many modules should be more resilient to damage in afferent connections, hence it is consistent with the conventional interpretation of the data on category specific impairment.

# References

[1] R. F. Hevner. More modules. *TINS*, 16(5):178, 1993.

[2] M. V. Tsodyks. Associative memory in neural networks with the hebbian learning rule. *Modern Physics Letters B*, 3(7):555–560, 1989.

[3] G. E. Hinton and T. Shallice. Lesioning at attractor network: investigations of acquired dyslexia. *Psychological Review*, 98(1):74–95, 1991.

[4] G. V. Jones. Deep dyslexia, imageability, and ease of predication. *Brain and Language*, 24:1–19, 1985.

[5] R. Lauro Grotto, S. Reich, and M. A. Virasoro. The computational role of conscious processing in a model of semantic memory. In *Proceedings of the IIAS Symposium on Cognition Computation and Consciousness*, 1994.

[6] P. A. Hetherington and L. M. Shapiro. Simulating hebb cell assemblies: the necessity for partitioned dendritic trees and a post-not-pre ltd rule. *Network*, 4:135–153, 1993.

[7] R. Yuste and D. W. Tank. Dendritic integration in mammalian neurons a century after cajal. *Neuron*, 16:701–716, 1996.